# Group Redundancy Measures Reveal Redundancy Reduction in the Auditory Pathway

**Gal Chechik**   **Amir Globerson**   **Naftali Tishby**
School of Computer Science and Engineering
and The Interdisciplinary Center for Neural Computation
Hebrew University of Jerusalem, Israel
*ggal@cs.huji.ac.il*

**Michael J. Anderson**      **Eric D. Young**
Department of Biomedical Engineering
Johns Hopkins University, Baltimore, MD, USA

**Israel Nelken**
Department of Physiology, Hadassah Medical School
and The Interdisciplinary Center for Neural Computation
Hebrew University of Jerusalem, Israel

## Abstract

The way groups of auditory neurons interact to code acoustic information is investigated using an information theoretic approach. We develop measures of redundancy among groups of neurons, and apply them to the study of collaborative coding efficiency in two processing stations in the auditory pathway: the inferior colliculus (IC) and the primary auditory cortex (AI). Under two schemes for the coding of the acoustic content, acoustic segments coding and stimulus identity coding, we show differences both in information content and group redundancies between IC and AI neurons. These results provide for the first time a direct evidence for redundancy reduction along the ascending auditory pathway, as has been hypothesized for theoretical considerations [Barlow 1959,2001]. The redundancy effects under the single-spikes coding scheme are significant only for groups larger than ten cells, and cannot be revealed with the redundancy measures that use only pairs of cells. The results suggest that the auditory system transforms low level representations that contain redundancies due to the statistical structure of natural stimuli, into a representation in which cortical neurons extract rare and independent component of complex acoustic signals, that are useful for auditory scene analysis.

# 1  Introduction

How do groups of sensory neurons interact to code information and how do these interactions change along the ascending sensory pathways ? According to the a common view, sensory systems are composed of a series of processing stations, representing more and more complex aspects of sensory inputs. The changes in representations of stimuli along the sensory pathway reflect the information processing performed by the system. Several computational principles that govern these changes were suggested, such as information maximization and redundancy reduction [2, 3, 11]. In order to investigate such changes in practice, it is necessary to develop methods to quantify information content and redundancies among groups of neurons, and trace these measures along the sensory pathway.

Interactions and high order correlations between neurons were mostly investigated within single brain areas on the level of pairs of cells (but also for larger groups of cells [9]) showing both synergistic and redundant interactions [8, 10, 21, 6, 7, 13]. The current study develops information theoretic redundancy measures for larger groups of neurons, focusing on the case of stimulus-conditioned independence. We then compare these measures in electro-physiological recordings from two auditory stations: the auditory mid-brain and the primary auditory cortex.

# 2  Redundancy measures for groups of neurons

To investigate high order correlations and interactions within groups of neurons we start by defining information measures for groups of cells and then develop information redundancy measures for such groups. The properties of these measures are then further discussed for the specific case of stimulus-conditioned independence.

Formally, the level of independence of two variables $X$ and $Y$ is commonly quantified by their *mutual information* (MI) [17, 5]. This well known quantity, now widely used in analysis of neural data, is defined by

$$I(X;Y) = D_{KL}[P(X,Y)||P(X)P(Y)] = \sum_{x,y} p(x,y)log\left(\frac{p(x,y)}{p(x)p(y)}\right) \qquad (1)$$

and measures how close the joint distribution $P(X,Y)$ is to the factorization by the marginal distributions $P(X)P(Y)$ ($D_{KL}$ is the Kullback Leiber divergence [5]).

For larger groups of cells, an important generalized measure quantifies the information that several variables provide about each other. This *multi information* measure [18] is defined by

$$\begin{aligned} I(X_1;...;X_n) &= D_{KL}[P(X_1,...,X_n)||P(X_1)...P(X_n)] = \qquad (2) \\ &= \sum_{x_1,...,x_n} p(x_1,...,x_n)log\left(\frac{p(x_1,...,x_n)}{p(x_1)...p(x_n)}\right) \quad . \end{aligned}$$

Similar to the *mutual information* case, the *multi information* measures how close the joint distribution is to the factorization by the marginals. It thus vanishes when variables are independent and is otherwise positive.

We now turn to develop measures for group redundancies. Consider first the simple case of a pair of neurons $(X_1, X_2)$ conveying information about the stimulus $S$. In this case, the redundancy-synergy index ([4, 7]) is defined by

$$RS_{pairs}(X_1, X_2, S) = I(X_1, X_2; S) - [I(X_1; S) + I(X_2; S)] \qquad (3)$$

Intuitively, $RS_{pairs}$ measures the amount of information on the stimulus $S$ gained by observing the joint distribution of both $X_1$ and $X_2$, as compared with observing the two cells independently. In the extreme case where $X_1 = X_2$, the two cells are completely redundant and provide the same information about the stimulus, yielding $RS_{pairs} = I(X_1, X_2; S) - I(X_1; S) - I(X_2; S) = -I(X_1; S)$, which is always non-positive. On the other hand, positive $RS_{pairs}$ values testify for synergistic interaction between $X_1$ and $X_2$ ([8, 7, 4]).

For larger groups of neurons, several different measures of redundancy-synergy may be considered, that encompass different levels of interactions. For example, one can quantify the residual information obtained from a group of $N$ neurons compared to all its $N-1$ subgroups. As with inclusion-exclusion calculations this measure takes the form of a telescopic sum: $RS_{N|N-1} = I(X^N; S) - \sum_{\{X^{N-1}\}} I(X^{N-1}; S) + ... + (-1)^{N-1} \sum_{\{X_i\}} I(X_i; S)$, where $\{X^k\}$ are all the subgroups of size $k$ out of the $N$ available neurons. Unfortunately, this measure involves $2^N$ information terms, making its calculation infeasible even for moderate $N$ values [1].

A different $RS$ measure quantifies the information embodied in the joint distribution of $N$ neurons compared to that provided by $N$ single independent neurons, and is defined by

$$RS_{N|1} = I(X_1, ..., X_N; S) - \sum_{i=1}^{N} I(X_i; S) \tag{4}$$

Interestingly, this synergy-redundancy measure may be rewritten as the difference between two multi-information terms

$$
\begin{aligned}
RS_{N|1} &= I(X_1, ..., X_N; S) - \sum_{i=1}^{N} I(X_i; S) = \\
&= H(X_1, ..., X_N) - H(X_1, ..., X_N|S) - \sum_{i=1}^{N} H(X_i) - H(X_i|S) = \\
&= I(X_1; ...; X_N|S) - I(X_1; ...; X_N)
\end{aligned} \tag{5}
$$

where $H(X) = -\sum_x p(x) log(p(x))$ is the entropy of $X$ [2]. We conclude that the index $RS_{N|1}$ can be separated into two terms: one that is always non-negative, and measures the coding synergy, and the second which is always non-positive and quantifies the redundancy. These two terms correspond to two types of interactions between neurons: The first type are *within-stimulus correlations* (sometimes termed noise correlations) that emerge from functional connections between neurons and contribute to synergy. The second type are *between stimulus correlations* (or *across stimulus correlations*) that reflect the fact that the cells have similar responses per stimulus, and contribute to redundancy. Being interested in the latter type of correlations, we limit the discussion to the redundancy term $-I(X_1; ...; X_N)$.

Formulating $RS_{N|1}$ as in equation 5 proves highly useful when neural activities are independent given the stimulus $P(\vec{X}|S) = \Pi_{i=1}^{N} P(X_i|S)$. In this case, the first (synergy) term vanishes, thus limiting neural interactions to the redundant

.

regime. More importantly, under the independence assumption we only have to estimate the marginal distributions $P(X_i|S = s)$ for each stimulus $s$ instead of the full distribution $P(\vec{X}|S = s)$. It thus allows to estimate an exponentially smaller number of parameters, which in our case of small sample sizes, provides more accurate information estimates. This approximation makes it possible to investigate redundancy among considerably larger groups of neurons than the 2-3 neuron groups considered previously in the literature.

How reasonable is the conditional-independence approximation ? It is a good approximation whenever neuronal activity is mostly determined by the presented stimulus and to a lesser extent by interactions with nearby neurons. A possible example is the high input regime of cortical neurons receiving thousands of inputs, where a single input has only a limited influence on the activity of the target cell. The experimental evidence in this regard is however mixed (see e.g.[9]). One should note however, that stimulus-conditioned independence is implicitly assumed in analysis of non-simultaneously recorded data.

To summarize, the stimulus-conditioned independence assumption limits interactions to the redundant regime, but allows to compare the extent of redundancy among large groups of cells in different brain areas.

## 3   Experimental Methods

To investigate redundancy in the auditory pathway, we analyze extracellular recordings from two brain areas of gas-anesthetized cats: 16 cells from the *Inferior Colliculus* (IC) - the third processing station of the ascending auditory pathway - and 19 cells from the *Primary Auditory Cortex* (AI) - the fifth station. Neural activity was recorded non-simultaneously from a total of 6 different animals responding to a set of complex natural and modified stimuli. Because cortical auditory neurons respond differently to simple and complex stimuli [12, 1], we refrain from using artificial over-simplified acoustic stimuli but instead use a set of stimuli based on bird vocalizations which contains complex 'real-life' acoustic features. A representative example is shown in figure 1.

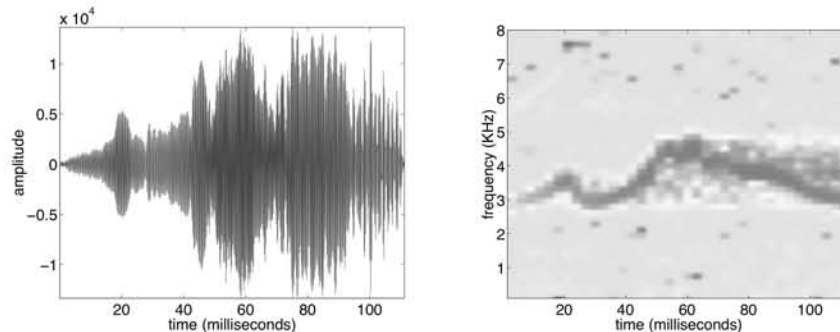

Figure 1: A representative stimulus containing a short bird vocalization recorded in a natural environment. The set of stimuli consisted of similar natural and modified recordings. **A.** Signal in time domain **B.** Signal in frequency domain.

## 4   Experimental Results

In practice, in order to estimate the information conveyed by neural activity from limited data, one must assume a decoding procedure, such as focusing on a simple statistic of the spike trains that encompasses some of its informative properties. In

this paper we consider two extreme cases: coding short acoustic segments with single spikes and coding the stimulus identity with spike counts in a long window. In addition, we estimated information and redundancy obtained with two other statistics. First, the latency of the first spike after stimulus onset, and secondly, a statistic which generalizes the counts statistics for a general renewal process [19]. These calculations yielded higher information content on average, but similar redundancies as presented below. Their detailed results will be reported elsewhere.

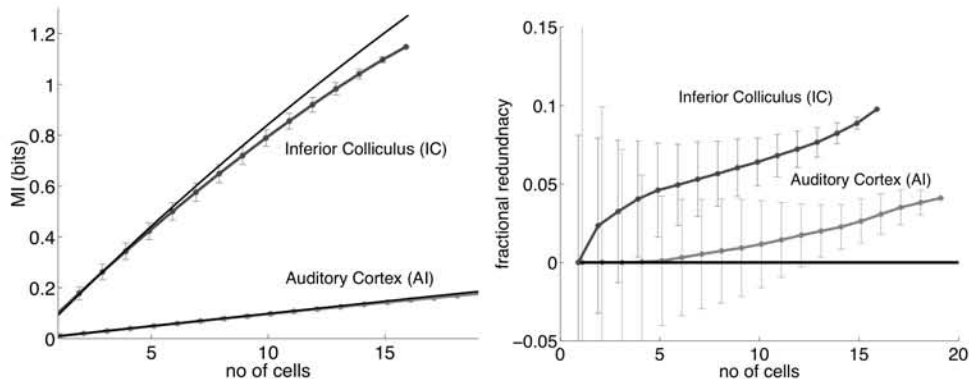

Figure 2: **A.** Information about stimulus frames as a function of number of cells. Information calculation was repeated for several subgroups of each size, and with several random seed initializations. The dark curve depicts the expected information provided by independent neurons (this expected curve is corrected for saturation effects [16] and is thus sub linear). The curved line depicts average information from joint distribution of sets of neurons $Mean[I(X_1, ...X_k; S)]$. All information estimations were corrected for small-samples bias by shuffling methods [14]. **B.** Fractional redundancy (difference of the mutual information from the expected baseline information divided by the baseline) as a function of number of neurons.

## 4.1 Coding acoustics with single spikes

The current section focuses on the relation between single spikes and short windows of the acoustic stimuli shortly preceding them (which we denote as *frames*). As the set of possible frames is very large and no frame actually repeats itself, we must first pre-process the stimuli to reduce frames dimensionality.

To this end, we first transformed the stimuli into the frequency domain (roughly approximating the cochlear transformation) and then extracted overlapping windows of 50 millisecond length, with 1 millisecond spacing. This set was clustered into 32 representatives, using a metric that groups together acoustic segments with the same spectro-temporal energy structure. This representation allowed us to estimate the joint distribution (under the stimulus-conditioned independence assumption) of cells' activity and stimuli, for groups of cells of different sizes. Figure 2A shows the mutual information between spikes and stimulus frames as a function of the number of cells for both AI and IC neurons. IC neurons convey high information but largely deviate from the information expected for independent neurons. On the other hand, AI neurons provide an order of magnitude less information than IC cells but their information sums almost linearly, as expected from independent neurons.

The difference between an information curve and its linear baseline measures the redundancy $RS_{N|1}$ of equation 5. Figure 2B presents the normalized redundancy as a function of number of cells, showing that IC cells are significantly more redundant

than AI cells.

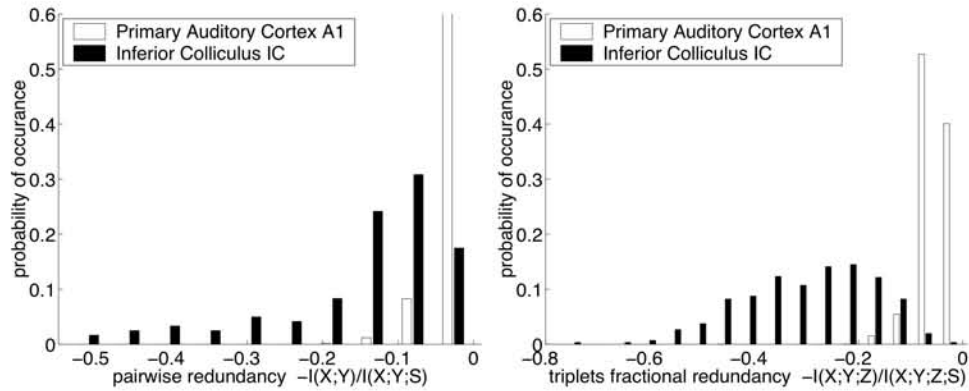

Figure 3: Distribution of pairs (**A.**) and triplets (**B.**) normalized redundancies. AI cells (light bars) are significantly more independent than IC cells (dark bars). Spike counts were collected over a window that maximizes mean single cells MI. Number of bins in counts-histogram was optimized separately for every cell. Information estimations were corrected for small-samples bias by shuffling methods [14].

## 4.2   Coding stimuli by spike counts

We now turn to investigate a second coding paradigm, and calculate the information conveyed by AI and IC **spike counts** about the **identity of the presented stimulus**. To this end, we calculate a histogram of spike counts and estimate the counts' distribution as obtained from repeated presentations of the stimuli.

The distribution of fractional redundancy in pairs of AI and IC neurons is presented in figure 3A, and that of triplets in figure 3B [3] . As in the case of coding with single spikes, single AI cells convey on average less information about the stimulus. However, they are also more independent, thus making it possible to gain more information from groups of neurons. IC neurons on the other hand, provide more information when considered separately but are more redundant.

As in the case of coding acoustics with single spikes, single IC cells provide more information than AI cells (data not shown) but this time AI cells convey half the information that IC cells provide, while they convey ten times less information than IC cells about acoustics. This suggests that AI cells poorly code the physical characteristics of the sound but convey information about its global properties. To illustrate the high information provided by both sets, we trained a neural network classifier that predicts the identity of the presented stimulus according to spike counts of a limited set of neurons. Figure 4 shows that both sets of neurons achieve considerable prediction accuracy, but IC neurons obtain average accuracy of more than 90 percent already with five cells, while the average prediction accuracy using cortical neurons rises continuously [4].

Figure 4. Prediction accuracy of stimulus identity as a function of number of IC (upper curve) and AI (lower curve) cells used by the classifier. Error bars denote standard deviation across several subgroups of the same size. For each subgroup, a one-hidden layer neural network was trained separately for each stimulus using some stimulus presentations as a training set and the rest for testing. Performance reported is for the testing set.

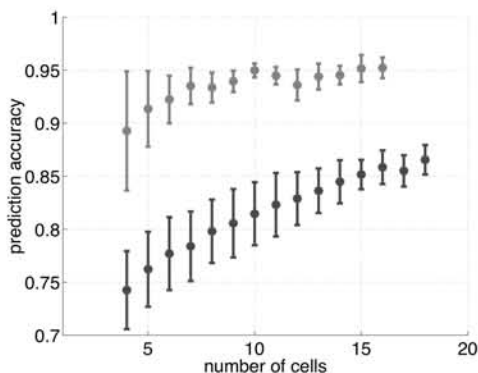

## 5 Discussion

We have developed information theoretic measures of redundancy among groups of neurons and applied them to investigate the collaborative coding efficiency in the auditory modality. Under two different coding paradigms, we show differences in both information content and group redundancies between IC and cortical auditory neurons. Single IC neurons carry more information about the presented stimulus, but are also more redundant. On the other hand, auditory cortical neurons carry less information but are more independent, thus allowing information to be summed almost linearly when considering groups of few tens of neurons. The results provide for the first time direct evidence for redundancy reduction along the ascending auditory pathway, as has been hypothesized by Barlow [2, 3]. The redundancy effects under the single-spikes coding paradigm are significant only for groups larger than ten cells, and cannot be revealed with the standard redundancy measures that use only pairs of cells.

Our results suggest that transformations leading to redundancy reduction are not limited to low level sensory processing (aimed to reduce redundancy in input statistics) but are applied even at cortical sensory stations. We suggest that an essential experimental prerequisite to reveal these effects is the use of complex acoustic stimuli whose processing occurs at high level processing stations.

The above findings are in agreement with the view that along the ascending sensory pathways, the number of neurons increase, their firing rates decrease, and neurons become tuned to more complex and independent features. Together, these suggest that the neural representation is mapped into a representation with higher effective dimensionality. Interestingly, recent advances in kernel-methods learning [20] have shown that nonlinear mapping into higher dimension and over-complete representations may be useful for learning of complex classifications. It is therefore possible that such mappings provide easier readout and more efficient learning in the brain.

### Acknowledgements

This work supported in part by a Human Frontier Science Project (HFSP) grant RG 0133/1998 and by a grant from the Israeli Ministry of Science.

## Footnotes

[1] Our results below suggest that some redundancy effects become significant only for groups larger than 10-15 cells.

[2] When comparing redundancy in different processing stations, one must consider the effects of the baseline information conveyed by single neurons. We thus use the normalized redundancy (compare with [15] p.315 and [4]) defined by $\hat{RS}_{N|1} = RS_{N|1}/I(X_1; ...; X_N; S)$

[3]Unlike the binary case of single spikes, the limited amount of data prevents a robust estimation of information from spike counts for more than triplets of cells.

[4]The probability of accurate prediction is exponentially related to the input-output mutual information, via the relation $P_{correct} = exp(-\text{missing nats})$ yielding $MI_{nats} = ln(\text{no. of stimuli}) + ln(P_{correct})$. Classification thus provides lower bounds on information content.

## References

[1] O. Bar-Yosef and I. Nelken. Responses of neurons in cat primary auditory cortex to bird chirps: Effects of temporal and spectral context. *J. Neuroscience*, in press, 2001.

[2] H.B. Barlow. Sensory mechanisms, the reduction of redundancy, and intelligence. In *Mechanisation of thought processes*, pages 535–539. Her Majesty's stationary office, London, 1959.

[3] H.B. Barlow. Redundancy reduction revisited. *Network: Computation in neural systems*, 12:241–253, 2001.

[4] N. Brenner, S.P. Strong, R. Koberle, R. de Ruyter van Steveninck, and W. Bialek. Synergy in a neural code. *Neural Computation*, 13(7):1531, 2000.

[5] T.M. Cover and J.A. Thomas. *The elements of information theory*. Plenum Press, New York, 1991.

[6] Y. Dan, J.M. Alonso, W.M. Usrey, and R.C. Reid. Coding of visual information by precisely correlated spikes in the lateral geniculate nucleus. *Nature Neuroscience*, 1(6):501–507, 1998.

[7] I. Gat and N. Tishby. Synergy and redundancy among brain cells of behaving monkeys. In M.S. Kearns, S.A. Solla, and D.A.Cohn, editors, *Advances in Neural Information Proceedings systems*, volume 11, Cambridge, MA, 1999. MIT Press.

[8] T.J. Gawne and B.J. Richmond. How independent are the messages carried by adjacent inferior temporal cortical neurons ? *J. Neurosci.*, 13(7):2758–2771, 1993.

[9] P.M. Gochin, M. Colombo, G. A. Dorfman, G.L. Gerstein, and C.G. Gross. Neural ensemble coding in inferior temporal cortex. *J. Neurophysiol.*, 71:2325–2337, 1994.

[10] M. Meister. Multineural codes in retinal signaling. *Proc. Natl. Acad. Sci.*, 93:609–614, 1996.

[11] J.P. Nadal, N. Brunel, and N. Parga. Nonlinear feedforward networks with stochastic outputs: infomax implies redundancy reduction. *Network: Computation in neural systems*, 9:207–217, 1998.

[12] I. Nelken, Y. Rotman, and O. Bar-Yosef. Specialization of the auditory system for the analysis of natural sounds. In J. Brugge and P.F. Poon, editors, *Central Auditory Processing and Neural Modeling*. Plenum, New York, 1997.

[13] S. Nirenberg, S.M. Carcieri, A.L. Jacobs, and P.E. Latham. Retinal ganglion cells act largely as independent encoders. *Nature*, 411:698–701, 2001.

[14] LM. Optican, T.J. Gawne, B.J. Richmond, and P.J. Joseph. Unbiased measures of transmitted information and channel capacity from multivariate neuronal data. *Biol. Cyber*, 65:305–310, 1991.

[15] E. T. Rolls and A. Treves. *Neural Networks and Brain Function*. Oxford Univ. Press, 1998.

[16] I. Samengo. Independent neurons representing a fintie set of stimuli: dependence of the mutual information on the number of units sampled. *Network: Comput. Neural Syst.*, 12:21–31, 2001.

[17] C.E. Shanon. A mathematical theory of communication. *The Bell systems technical journal*, 27:379–423,623–656, 1948.

[18] M. Studeny and J. Vejnarova. The multiinformation function as a tool for measuring stochastic dependence. In M.I. Jordan, editor, *Learning in Graphical Models*, pages 261–297. Dordrecht: Kluwer, 1998.

[19] C. van Vreeswijk. Information trasmission with renewal neurons. In J.M. Bower, editor, *Computational Neuroscience: Trends in Research*. Elsevier Press, 2001.

[20] V.N. Vapnik. *The nature of statistical learning theory*. Springer-Verlag, Berlin, 1995.

[21] DK. Warland, P. Reinagel, and M. Meister. Decoding visual information from a population of retinal ganglion cells. *J. Neurophysiol.*, 78:2336–2350, 1997.
